# Nearest-Neighbor-Based Active Learning for Rare Category Detection

**Jingrui He**
School of Computer Science
Carnegie Mellon University
jingruih@cs.cmu.edu

**Jaime Carbonell**
School of Computer Science
Carnegie Mellon University
jgc@cs.cmu.edu

## Abstract

Rare category detection is an open challenge for active learning, especially in the *de-novo* case (no labeled examples), but of significant practical importance for data mining - e.g. detecting new financial transaction fraud patterns, where normal legitimate transactions dominate. This paper develops a new method for detecting an instance of each minority class via an unsupervised local-density-differential sampling strategy. Essentially a variable-scale nearest neighbor process is used to optimize the probability of sampling tightly-grouped minority classes, subject to a local smoothness assumption of the majority class. Results on both synthetic and real data sets are very positive, detecting each minority class with only a fraction of the actively sampled points required by random sampling and by Pelleg's Interleave method, the prior best technique in the sparse literature on this topic.

## 1 Introduction

In many real world problems, the proportion of data points in different classes is highly skewed: some classes dominate the data set (majority classes), and the remaining classes may have only a few examples (minority classes). However, it is very important to detect examples from the minority classes via active learning. For example, in fraud detection tasks, most of the records correspond to normal transactions, and yet once we identify a new type of fraud transaction, we are well on our way to stopping similar future fraud transactions [2]. Another example is in astronomy. Most of the objects in sky survey images are explainable by current theories and models. Only 0.001% of the objects are truly beyond the scope of current science and may lead to new discoveries [8]. Rare category detection is also a bottleneck in reducing the sampling complexity of active learning [1, 5]. The difference between rare category detection and outlier detection is that: in rare category detection, the examples from one or more minority classes are often self-similar, potentially forming compact clusters, while in outlier detection, the outliers are typically scattered.

Currently, only a few methods have been proposed to address this challenge. For example, in [8], the authors assumed a mixture model to fit the data, and selected examples for labeling according to different criteria; in [6], the authors proposed a generic consistency algorithm, and proved upper bounds and lower bounds for this algorithm in some specific situations. Most of the existing methods require that the majority classes and the minority classes be separable or work best in the separable case. However, in real applications, the support regions of the majority and minority classes often overlap, which affects negatively the performance of these methods.

In this paper, we propose a novel method for rare category detection in the context of active learning. We typically start *de-novo*, no category labels, though our algorithm makes no such assumption. Different from existing methods, we aim to solve the hard case, i.e. we do not assume separability or near-separability of the classes. Intuitively, the method makes use of nearest neighbors to measure local density around each example. In each iteration, the algorithm selects an example with the

maximum change in local density on a certain scale, and asks the oracle for its label. The method stops once it has found at least one example from each class (given the knowledge of the number of classes). When the minority classes form compact clusters and the majority class distribution is locally smooth, the method will select examples both on the boundary and in the interior of the minority classes, and is proved to be effective theoretically. Experimental results on both synthetic and real data sets show the superiority of our method over existing methods.

The rest of the paper is organized as follows. In Section 2, we introduce our method and provide theoretical justification, first for binary classes and then for multiple classes. Section 3 gives experimental results. Finally, we conclude the paper in Section 4.

## 2 Rare category detection

### 2.1 Problem definition

Given a set of unlabeled examples $S = \{x_1, \ldots, x_n\}$, $x_i \in \mathbb{R}^d$, which come from $m$ distinct classes, i.e. $y_i \in \{1, \ldots, m\}$, the goal is to find at least one example from each class by requesting as few total labels as possible. For the sake of simplicity, assume that there is only one majority class, which corresponds to $y_i = 1$, and all the other classes are minority classes.

### 2.2 Rare category detection for the binary case

First let us focus on the simplest case where $m = 2$, and $\Pr[y_i = 1] \gg \Pr[y_i = 2] = p$, i.e. $p \ll 1$. Here, we assume that we have an estimate of the value of $p$ a priori. Next, we introduce our method for rare category detection based on nearest neighbors, which is presented in Algorithm 1. The basic idea is to find maximum changes in local density, which might indicate the location of a rare category.

The algorithm works as follows. Given the unlabeled set $S$ and the prior of the minority class $p$, we first estimate the number $K$ of minority class examples in $S$. Then, for each example, we record its distance from the $K^{\text{th}}$ nearest neighbor, which could be realized by kd-trees [7]. The minimum distance over all the examples is assigned to $r'$. Next, we draw a hyper-ball centered at each example with radius $r'$, and count the number of examples enclosed by this hyper-ball, which is denoted as $n_i$. $n_i$ is roughly in proportion to the local density. To measure the change of local density around a certain point $x_i$, in each iteration of Step 3, we subtract $n_j$ of neighboring points from $n_i$, and let the maximum value be the score of $x_i$. The example with the maximum score is selected for labeling by the oracle. If the example is from the minority class, stop the iteration; otherwise, enlarge the neighborhood where the scores of the examples are re-calculated and continue.

Before giving theoretical justification, here, we give an intuitive explanation of why the algorithm works. Assume that the minority class is concentrated in a small region and the probability distribution function (pdf) of the majority class is locally smooth. Firstly, since the support region of the minority class is very small, it is important to find its scale. The $r'$ value obtained in Step 1 will be used to calculate the local density $n_i$. Since $r'$ is based on the minimum $K^{\text{th}}$ nearest neighbor distance, it is never too large to smooth out changes of local density, and thus it is a good measure of the scale. Secondly, the score of a certain point, corresponding to the change in local density, is the maximum of the difference in local density between this point and all of its neighboring points. In this way, we are not only able to select points on the boundary of the minority class, but also points in the interior, given that the region is small. Finally, by gradually enlarging the neighborhood where the scores are calculated, we can further explore the interior of the support region, and increase our chance of finding a minority class example.

### 2.3 Correctness

In this subsection, we prove that if the minority class is concentrated in a small region and the pdf of the majority class is locally smooth, the proposed algorithm will repeatedly sample in the region where minority class examples occur with high probability.

Let $f_1(x)$ and $f_2(x)$ denote the pdf of the majority and minority classes respectively, where $x \in \mathbb{R}^d$. To be precise, we make the following assumptions.

---
**Algorithm 1** Nearest-Neighbor-Based Rare Category Detection for the Binary Case (NNDB)
---
**Require:** $S, p$
1: Let $K = np$. For each example, calculate the distance to its $K^{\text{th}}$ nearest neighbor. Set $r'$ to be the minimum value among all the examples.
2: $\forall x_i \in S$, let $NN(x_i, r') = \{x | x \in S, \|x - x_i\| \leq r'\}$, and $n_i = |NN(x_i, r')|$.
3: **for** $t = 1 : n$ **do**
4: $\quad \forall x_i \in S$, if $x_i$ has not been selected, then $s_i = \max\limits_{x_j \in NN(x_i, tr')} (n_i - n_j)$; otherwise, $s_i = -\infty$.
5: $\quad$ Query $x = \arg\max_{x_i \in S} s_i$.
6: $\quad$ If the label of $x$ is 2, break.
7: **end for**
---

**Assumptions**

1. $f_2(x)$ is uniform within a hyper-ball $B$ of radius $r$ centered at $b$, i.e. $f_2(x) = \frac{1}{V(r)}$, if $x \in B$; and 0 otherwise, where $V(r) \propto r^d$ is the volume of $B$.

2. $f_1(x)$ is bounded and positive in $B^1$, i.e. $f_1(x) \geq \frac{c_1 p}{(1-p)V(r)}$, $\forall x \in B$ and $f_1(x) \leq \frac{c_2 p}{(1-p)V(r)}$, $\forall x \in \mathbb{R}^d$, where $c_1, c_2 > 0$ are two constants.

With the above assumptions, we have the following claim and theorem. Note that variants of the following proof apply if we assume a different minority class distribution, such as a tight Gaussian.

**Claim 1.** $\forall \epsilon, \delta > 0$, if $n \geq \max\{\frac{1}{2c_1^2 p^2} \log \frac{3}{\delta}, \frac{1}{2(1-2^{-d})^2 p^2} \log \frac{3}{\delta}, \frac{1}{\epsilon^4 V(\frac{r_2}{2})^4} \log \frac{3}{\delta}\}$, where $r_2 = \frac{r}{(1+c_2)^{\frac{1}{d}}}$, and $V(\frac{r_2}{2})$ is the volume of a hyper-ball with radius $\frac{r_2}{2}$, then with probability at least $1 - \delta$, $\frac{r_2}{2} \leq r' \leq r$ and $|\frac{n_i}{n} - E(\frac{n_i}{n})| \leq \epsilon V(r')$, $1 \leq i \leq n$, where $V(r')$ is the volume of a hyper-ball with radius $r'$.

*Proof.* First, notice that the expected proportion of points falling inside $B$, $E(\frac{|NN(b,r)|}{n}) \geq (c_1 + 1)p$, and that the maximum expected proportion of points falling inside any hyper-ball of radius $\frac{r_2}{2}$, $\max\limits_{x \in \mathbb{R}^d}[E(\frac{|NN(x, \frac{r_2}{2})|}{n})] \leq 2^{-d} p$. Then

$$\Pr[r' > r \text{ or } r' < \frac{r_2}{2} \text{ or } \exists x_i \in S \text{ s.t. } |\frac{n_i}{n} - E(\frac{n_i}{n})| > \epsilon V(r')]$$

$$\leq \Pr[r' > r] + \Pr[r' < \frac{r_2}{2}] + \Pr[r' \geq \frac{r_2}{2} \text{ and } \exists x_i \in S \text{ s.t. } |\frac{n_i}{n} - E(\frac{n_i}{n})| > \epsilon V(r')]$$

$$\leq \Pr[|NN(b,r)| < K] + \Pr[\max\limits_{x \in \mathbb{R}^d} |NN(x, \frac{r_2}{2})| > K] + n\Pr[|\frac{n_i}{n} - E(\frac{n_i}{n})| > \epsilon V(r')|r' \geq \frac{r_2}{2}]$$

$$= \Pr[|\frac{NN(b,r)}{n}| < p] + \Pr[\max\limits_{x \in \mathbb{R}^d} |\frac{NN(x, \frac{r_2}{2})}{n}| > p] + n\Pr[|\frac{n_i}{n} - E(\frac{n_i}{n})| > \epsilon V(r')|r' \geq \frac{r_2}{2}]$$

$$\leq e^{-2nc_1^2 p^2} + e^{-2n(1-2^{-d})^2 p^2} + 2ne^{-2n\epsilon^2 V(r')^2}$$

where the last inequality is based on Hoeffding bound.

Let $e^{-2nc_1^2 p^2} \leq \frac{\delta}{3}$, $e^{-2n(1-2^{-d})^2 p^2} \leq \frac{\delta}{3}$ and $2ne^{-2n\epsilon^2 V(r')} \leq 2ne^{-2n\epsilon^2 V(\frac{r_2}{2})^2} \leq \frac{\delta}{3}$, we obtain $n \geq \frac{1}{2c_1^2 p^2} \log \frac{3}{\delta}$, $n \geq \frac{1}{2(1-2^{-d})^2 p^2} \log \frac{3}{\delta}$, and $n \geq \frac{1}{\epsilon^4 V(\frac{r_2}{2})^4} \log \frac{3}{\delta}$. ∎

Based on Claim 1, we get the following theorem, which shows the effectiveness of the proposed method.

**Main Theorem.** If

1. Let $B^2$ be the hyper-ball centered at $b$ with radius $2r$. The minimum distance between the points inside $B$ and the ones outside $B^2$ is not too large, i.e. $\min\{\|x_i - x_j\| | x_i, x_j \in S, \|x_i - b\| \leq r, \|x_j - b\| > 2r\} \leq \alpha$, where $\alpha$ is a positive parameter.

2. $f_1(x)$ is locally smooth, i.e. $\forall x, y \in \mathbb{R}^d, |f_1(x) - f_1(y)| \leq \frac{\beta \|x-y\|}{\alpha}$, where $\beta \leq \frac{p^2 OV(\frac{r_2}{2}, r)}{2^{d+1} V(r)^2}$ and $OV(\frac{r_2}{2}, r)$ is the volume of the overlapping region of two hyper-balls: one is of radius $r$, the other one is of radius $\frac{r_2}{2}$, and its center is on the sphere of the bigger one.

3. The number of examples is sufficiently large,
   i.e. $n \geq \max\{\frac{1}{2c_1^2 p^2} \log \frac{3}{\delta}, \frac{1}{2(1-2^{-d})^2 p^2} \log \frac{3}{\delta}, \frac{1}{(1-p)^4 \beta^4 V(\frac{r_2}{2})^4} \log \frac{3}{\delta}\}$.

then with probability at least $1 - \delta$, after $\lceil \frac{2\alpha}{r_2} \rceil$ iterations, NNDB will query at least one example whose probability of coming from the minority class is at least $\frac{1}{3}$, and it will continue querying such examples until the $\lfloor (\frac{2^d}{p(1-p)} - 2) \cdot \frac{\alpha}{r} \rfloor^{\text{th}}$ iteration.

*Proof.* Based on Claim 1, using condition 3, if the number of examples is sufficiently large, then with probability at least $1 - \delta$, $\frac{r_2}{2} \leq r' \leq r$ and $|\frac{n_i}{n} - E(\frac{n_i}{n})| \leq (1-p)\beta V(r')$, $1 \leq i \leq n$. According to condition 2, $\forall x_i, x_j \in S$ s.t. $\|x_i - b\| > 2r$, $\|x_j - b\| > 2r$ and $\|x_i - x_j\| \leq \alpha$, $E(\frac{n_i}{n})$ and $E(\frac{n_j}{n})$ will not be affected by the minority class, and $|E(\frac{n_i}{n}) - E(\frac{n_j}{n})| \leq (1-p)\beta V(r') \leq (1-p)\beta V(r)$. Note that $\alpha$ is always bigger than $r$. Based on the above inequalities, we have

$$|\frac{n_i}{n} - \frac{n_j}{n}| \leq |\frac{n_i}{n} - E(\frac{n_i}{n})| + |\frac{n_j}{n} - E(\frac{n_j}{n})| + |E(\frac{n_i}{n}) - E(\frac{n_j}{n})| \leq 3(1-p)\beta V(r) \quad (1)$$

From inequality (1), it is not hard to see that $\forall x_i, x_j \in S$, s.t. $\|x_i - b\| > 2r$ and $\|x_i - x_j\| \leq \alpha$, $\frac{n_i}{n} - \frac{n_j}{n} \leq 3(1-p)\beta V(r)$, i.e. when $tr' = \alpha$,

$$\frac{s_i}{n} \leq 3(1-p)\beta V(r) \quad (2)$$

This is because if $\|x_j - b\| \leq 2r$, the minority class may also contribute to $\frac{n_j}{n}$, and thus the score may be even smaller.

On the other hand, based on condition 1, there exist two points $x_k, x_l \in S$, s.t. $\|x_k - b\| \leq r$, $\|x_l - b\| > 2r$, and $\|x_k - x_l\| \leq \alpha$. Since the contribution of the minority class to $E(\frac{n_k}{n})$ is at least $\frac{p \cdot OV(\frac{r_2}{2}, r)}{V(r)}$, so $E(\frac{n_k}{n}) - E(\frac{n_l}{n}) \geq \frac{p \cdot OV(\frac{r_2}{2}, r)}{V(r)} - (1-p)\beta V(r') \geq \frac{p \cdot OV(\frac{r_2}{2}, r)}{V(r)} - (1-p)\beta V(r)$. Since for any example $x_i \in S$, we have $|\frac{n_i}{n} - E(\frac{n_i}{n})| \leq (1-p)\beta V(r') \leq (1-p)\beta V(r)$, therefore

$$\frac{n_k}{n} - \frac{n_l}{n} \geq \frac{p \cdot OV(\frac{r_2}{2}, r)}{V(r)} - 3(1-p)\beta V(r) \geq \frac{p \cdot OV(\frac{r_2}{2}, r)}{V(r)} - \frac{3(1-p)p^2 \cdot OV(\frac{r_2}{2}, r)}{2^{d+1} V(r)}$$

Since $p$ is very small, $p \gg \frac{3(1-p)p^2}{2^{d+1}}$; therefore, $\frac{n_k}{n} - \frac{n_l}{n} > 3(1-p)\beta V(r)$, i.e. when $tr' = \alpha$,

$$\frac{s_k}{n} > 3(1-p)\beta V(r) \quad (3)$$

In Step 4 of the proposed method, we gradually enlarge the neighborhood to calculate the change of local density. When $tr' = \alpha$, based on inequalities (2) and (3), $\forall x_i \in S$, $\|x_i - b\| > 2r$, we have $s_k > s_i$. Therefore, in this round of iteration, we will pick an example from $B^2$. In order for $tr'$ to be equal to $\alpha$, the value of $t$ would be $\lceil \frac{\alpha}{r'} \rceil \leq \lceil \frac{2\alpha}{r_2} \rceil$.

If we further increase $t$ so that $tr' = c\alpha$, where $c > 1$, we have the following conclusion: $\forall x_i, x_j \in S$, s.t. $\|x_i - b\| > 2r$ and $\|x_i - x_j\| \leq c\alpha$, $\frac{n_i}{n} - \frac{n_j}{n} \leq (c+2)(1-p)\beta V(r)$, i.e. $\frac{s_i}{n} \leq (c+2)(1-p)\beta V(r)$. As long as $p \geq \frac{(c+2)(1-p)p^2}{2^d}$, i.e. $c \leq \frac{2^d}{p(1-p)} - 2$, then $\forall x_i \in S$, $\|x_i - b\| > 2r$, $s_k > s_i$, and we will pick examples from $B^2$. Since $r' \leq r$, the method will continue querying examples in $B^2$ until the $\lfloor (\frac{2^d}{p(1-p)} - 2) \cdot \frac{\alpha}{r} \rfloor^{\text{th}}$ iteration.

Finally, we show that the probability of picking a minority class example from $B^2$ is at least $\frac{1}{3}$. To this end, we need to calculate the maximum probability mass of the majority class within $B^2$. Consider the case where the maximum value of $f_1(x)$ occurs at $b$, and this pdf decreases by $\beta$ every time $x$ moves away from $b$ in the direction of the radius by $\alpha$, i.e. the shape of $f_1(x)$ is a cone in $(d+1)$ dimensional space. Since $f_1(x)$ must integrate to 1, i.e. $V(\frac{\alpha f_1(b)}{\beta}) \cdot \frac{f_1(b)}{d+1} = 1$, where $V(\frac{\alpha f_1(b)}{\beta})$ is the volume of a hyper-ball with radius $\frac{\alpha f_1(b)}{\beta}$, we have $f_1(b) = (\frac{d+1}{V(\alpha)})^{\frac{1}{d+1}} \beta^{\frac{d}{d+1}}$.

Therefore, the probability mass of the majority class within $B^2$ is:

$$V(2r)(f_1(b) - \frac{2r}{\alpha}\beta) + \frac{2r}{\alpha}\frac{\beta}{d+1}V(2r) < V(2r)f_1(b)$$

$$= V(2r)(\frac{d+1}{V(\alpha)})^{\frac{1}{d+1}}\beta^{\frac{d}{d+1}} = 2^d\frac{V(r)}{V(\alpha)^{\frac{1}{d+1}}}(d+1)^{\frac{1}{d+1}}\beta^{\frac{d}{d+1}}$$

$$< (d+1)^{\frac{1}{d+1}}(2^{d+1}V(r)\beta)^{\frac{d}{d+1}} \le (d+1)^{\frac{1}{d+1}}(\frac{p^2 \cdot OV(\frac{r_2}{2},r)}{V(r)})^{\frac{d}{d+1}} < 2p$$

where $V(2r)$ is the volume of a hyper-ball with radius $2r$. Therefore, if we select a point at random from $B^2$, the probability that this point is from the minority class is at least $\frac{p}{p+(1-p)\cdot 2p} \ge \frac{p}{p+2p} = \frac{1}{3}$. ∎

## 2.4 Rare category detection for multiple classes

In subsection 2.2, we have discussed rare category detection for the binary case. In this subsection, we focus on the case where $m > 2$. To be specific, let $p_1, \ldots, p_m$ be the priors of the $m$ classes, and $p_1 \gg p_i$, $i \ne 1$. Our goal is to use as few label requests as possible to find at least one example from each class.

The method proposed in subsection 2.2 can be easily generalized to multiple classes, which is presented in Algorithm 2. In this algorithm, we are given the priors of all the minority classes. Using each $p_i$, we estimate the number $K_i$ of examples from this class, and calculate the corresponding $r_i'$ value in the same manner as NNDB. Then, we calculate the local density at each example based on different scales $r_i'$. In the outer loop of Step 9, we calculate the $r'$ value which is the minimum of all the $r_i'$ whose corresponding classes have not been discovered yet and its index. In the inner loop of Step 11, we gradually enlarge the neighborhood to calculate the score of each example. This is the same as NNDB, except that we preclude the examples that are within a certain distance of any selected example from being selected. This heuristic is to avoid repeatedly selecting examples from the same discovered class. The inner loop stops when we find an example from an undiscovered class. Then we will update the $r'$ value and resume the inner loop. If the minority classes form compact clusters and are far apart from each other, NNDM is able to detect examples from each minority class with a small number of label requests.

---

**Algorithm 2** Nearest-Neighbor-Based Rare Category Detection for Multiple Classes (NNDM)

---

**Require:** $S, p_2, \ldots, p_m$

1: **for** $i = 2 : m$ **do**
2:      Let $K_i = np_i$.
3:      For each example, calculate the distance between this example and its $K_i^{\text{th}}$ nearest neighbor. Set $r_i'$ to be the minimum value among all the examples.
4: **end for**
5: Let $r_1' = \max_{i=2}^m r_i'$.
6: **for** $i = 1 : m$ **do**
7:      $\forall x_j \in S$, let $NN(x_j, r_i') = \{x | x \in S, \|x - x_j\| \le r_i'\}$, and $n_j^i = |NN(x_j, r_i')|$.
8: **end for**
9: **while** not all the classes have been discovered **do**
10:      Let $r' = \min\{r_i' | 1 \le i \le m, \text{and class } i \text{ has not been discovered}\}$, and $s$ be the corresponding index, i.e. $r' = r_s'$.
11:      **for** $t = 1 : n$ **do**
12:          for each $x_i$ that has been selected and labeled $y_i$, $\forall x \in S$, s.t. $\|x - x_i\| \le r_{y_i}'$, $s_i = -\infty$; for all the other examples, $s_i = \max_{x_j \in NN(x_i, tr')}(n_i^s - n_j^s)$.
13:          Query $x = \arg\max_{x_i \in S} s_i$.
14:          If $x$ belongs to a class that has not been discovered, break.
15:      **end for**
16: **end while**

---

In NNDB and NNDM, we need the priors of the minority classes as the input. As we will see in the next section, our algorithms are robust against small perturbations in the priors.

# 3 Experimental results

In this section, we compare our methods (NNDB and NNDM) with the best method proposed in [8] (Interleave) and random sampling (RS) on both synthetic and real data sets. In Interleave, we use the number of classes as the number of components in the mixture model. For both Interleave and RS, we run the experiment multiple times and report the average results.

## 3.1 Synthetic data sets

Figure 1(a) shows a synthetic data set where the pdf of the majority class is Gaussian and the pdf of the minority class is uniform within a small hyper-ball. There are 1000 examples from the majority class and only 10 examples from the minority class. Using Interleave, we need to label 35 examples, using RS, we need to label 101 examples, and using NNDB, we only need to label 3 examples in order to sample one from the minority class, which are denoted as 'x' in Figure 1(b). Notice that the first 2 examples that NNDB selects are not from the correct region. This is because the number of examples from the minority class is very small, and the local density may be affected by the randomness in the data.

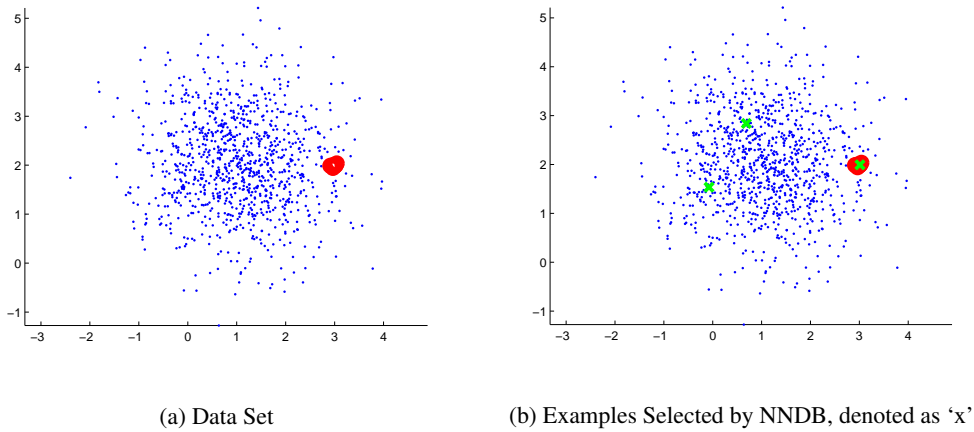

(a) Data Set        (b) Examples Selected by NNDB, denoted as 'x'

Figure 1: Synthetic Data Set 1.

In Figure 2(a), the X-shaped data consisting of 3000 examples correspond to the majority class, and the four characters 'NIPS' correspond to four minority classes, which consist of 138, 79, 118, and 206 examples respectively. Using Interleave, we need to label 1190 examples, using RS, we need to label 83 examples, and using NNDM, we only need to label 5 examples in order to get one from each of the minority classes, which are denoted as 'x' in Figure 2(b). Notice that in this example, Interleave is even worse than RS. This might be because some minority classes are located in the region where the density of the majority class is not negligible, and thus may be 'explained' by the majority-class mixture-model component.

## 3.2 Real data sets

In this subsection, we compare different methods on two real data sets: Abalone [3] and Shuttle [4]. The first data set consists of 4177 examples, described by 7 dimensional features. The examples come from 20 classes: the proportion of the largest class is 16.50%, and the proportion of the smallest class is 0.34%. For the second data set, we sub-sample the original training set to produce a smaller data set with 4515 examples, described by 9 dimensional features. The examples come from 7 classes: the proportion of the largest class is 75.53%, and the proportion of the smallest class is 0.13%.

The comparison results are shown in Figure 3(a) and Figure 3(b) respectively. From these figures, we can see that NNDM is significantly better than Interleave and RS: with Abalone data set, to find

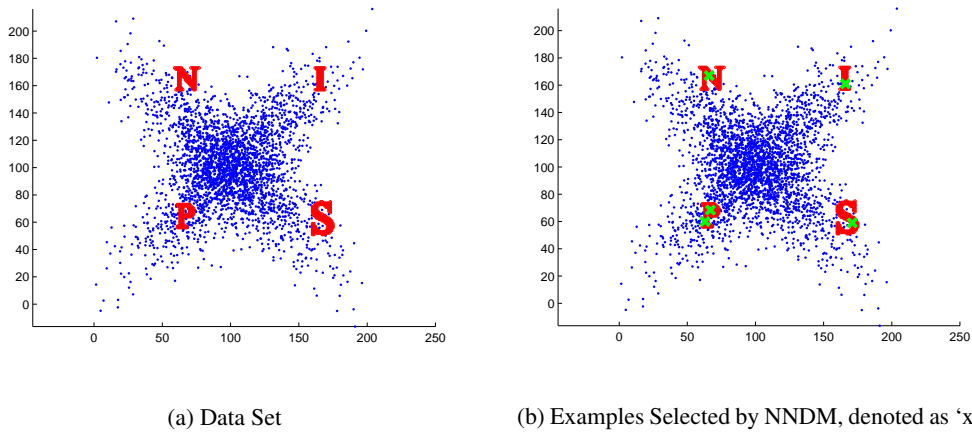

(a) Data Set  (b) Examples Selected by NNDM, denoted as 'x'

Figure 2: Synthetic Data Set 2.

all the classes, Interleave needs 280 label requests, RS needs 483 label requests, and NNDM only needs 125 label requests; with Shuttle data set, to find all the classes, Interleave needs 140 label requests, RS needs 512 label requests, and NNDM only needs 87 label requests. This is because as the number of components becomes larger, the mixture model generated by Interleave is less reliable due to the lack of labeled examples, thus we need to select more examples. Furthermore, the majority and minority classes may not be near-separable, which is a disaster for Interleave. On the other hand, NNDM does not assume a generative model for the data, and only focuses on the change in local density, which is more effective on the two data sets.

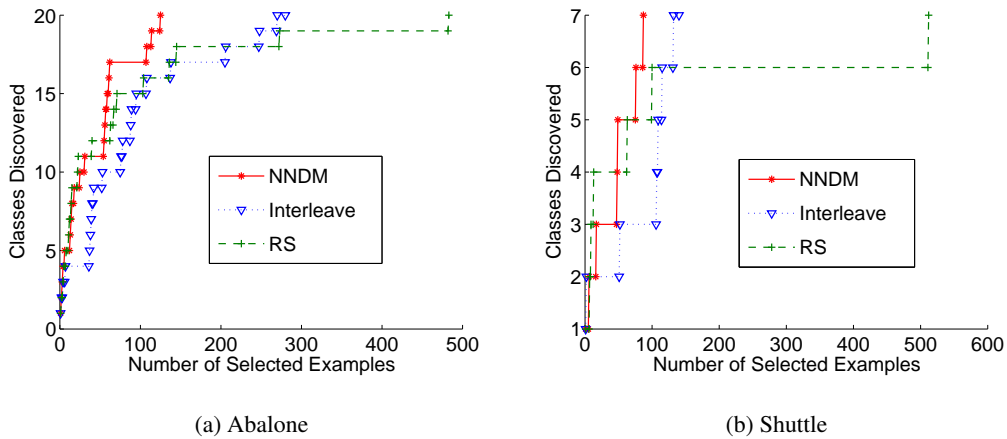

(a) Abalone  (b) Shuttle

Figure 3: Learning Curves for Real Data Sets

## 3.3 Imprecise priors

The proposed algorithms need the priors of the minority classes as input. In this subsection, we test the robustness of NNDM against modest mis-estimations of the class priors. The performance of NNDB is similar to NNDM, so we omit the results here. In the experiments, we use the same data sets as in subsection 3.2, and add/subtract 5%, 10%, and 20% from the true priors of the minority classes. The results are shown in Figure 4. From these figures, we can see that NNDM is very robust to small perturbations in the priors. For example, with Abalone data set, if we subtract 10% from the true priors, only one more label request is needed in order to find all the classes.

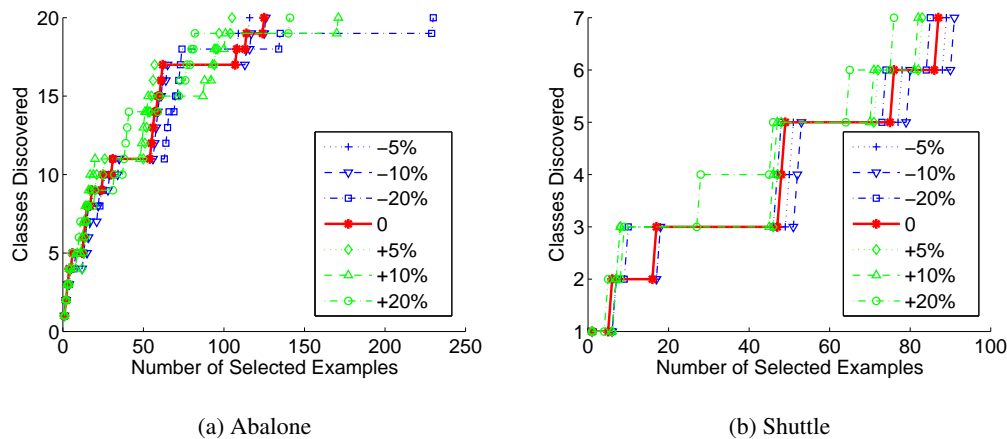

|                          |                       |
|:------------------------:|:---------------------:|
| (a) Abalone              | (b) Shuttle           |

Figure 4: Robustness Study

## 4 Conclusion

In this paper, we have proposed a novel method for rare category detection, useful for *de-novo* active learning in serious applications. Different from existing methods, our method does not rely on the assumption that the data is near-separable. It works by selecting examples corresponding to regions with the maximum change in local density, and depending on scaling, it will select class-boundary or class-internal samples of minority classes. The method could be scaled up using kd-trees [7]. The effectiveness of the proposed method is guaranteed by theoretical justification, and its superiority over existing methods is demonstrated by extensive experimental results on both synthetic and real data sets. Moreover, it is very robust to modest perturbations in estimating true class priors.

## Acknowledgments

This paper is based on work in part supported by the Defense Advanced Research Projects Agency (DARPA) under contract number NBCHD030010.

## Footnotes

[1] Notice that here we are only dealing with the hard case where $f_1(x)$ is positive within $B$. In the separable case where the support regions of the two classes do not overlap, we can use other methods to detect the minority class, such as the one proposed in [8].

## References

[1] M. Balcan, A. Beygelzimer, and J. Langford. Agnostic active learning. In *Proc. of the 23rd Int. Conf. on Machine Learning*, pages 65–72, 2006.

[2] S. Bay, K. Kumaraswamy, M. Anderle, R. Kumar, and D. Steier. Large scale detection of irregularities in accounting data. In *Proc. of the 6th Int. Conf. on Data Mining*, pages 75–86, 2006.

[3] C. Blake and C. Merz. Uci repository of machine learning databases. In *http://www.ics.uci.edu/ machine/MLRepository.html*, 1998.

[4] P. Brazdil and J. Gama. Statlog repository. In *http://www.niaad.liacc.up.pt/old/statlog/datasets/shuttle/shuttle.doc.html*, 1991.

[5] S. Dasgupta. Coarse sample complexity bounds for active learning. In *Advances in Neural Information Processing Systems 19*, 2005.

[6] S. Fine and Y. Mansour. Active sampling for multiple output identification. In *The 19th Annual Conf. on Learning Theory*, pages 620–634, 2006.

[7] A. Moore. A tutorial on kd-trees. Technical report, University of Cambridge Computer Laboratory, 1991.

[8] D. Pelleg and A. Moore. Active learning for anomaly and rare-category detection. In *Advances in Neural Information Processing Systems 18*, 2004.

